# Reinforcement Learning Applied to Linear Quadratic Regulation

**Steven J. Bradtke**
Computer Science Department
University of Massachusetts
Amherst, MA 01003
bradtke@cs.umass.edu

## Abstract

Recent research on reinforcement learning has focused on algorithms based on the principles of Dynamic Programming (DP). One of the most promising areas of application for these algorithms is the control of dynamical systems, and some impressive results have been achieved. However, there are significant gaps between practice and theory. In particular, there are no convergence proofs for problems with continuous state and action spaces, or for systems involving non-linear function approximators (such as multilayer perceptrons). This paper presents research applying DP-based reinforcement learning theory to Linear Quadratic Regulation (LQR), an important class of control problems involving continuous state and action spaces and requiring a simple type of non-linear function approximator. We describe an algorithm based on $Q$-learning that is proven to converge to the optimal controller for a large class of LQR problems. We also describe a slightly different algorithm that is only *locally* convergent to the optimal $Q$-function, demonstrating one of the possible pitfalls of using a non-linear function approximator with DP-based learning.

## 1 INTRODUCTION

Recent research on reinforcement learning has focused on algorithms based on the principles of Dynamic Programming. Some of the DP-based reinforcement learning

algorithms that have been described are Sutton's Temporal Differences methods (Sutton, 1988), Watkins' $Q$-learning (Watkins, 1989), and Werbos' Heuristic Dynamic Programming (Werbos, 1987). However, there are few convergence results for DP-based reinforcement learning algorithms, and these are limited to discrete time, finite-state systems, with either lookup-tables or linear function approximators. Watkins and Dayan (1992) show that the $Q$-learning algorithm converges, under appropriate conditions, to the optimal $Q$-function for finite-state Markovian decision tasks, where the $Q$-function is represented by a lookup-table. Sutton (1988) and Dayan (1992) show that the linear TD($\lambda$) learning rule, when applied to Markovian decision tasks where the states are represented by a linearly independent set of feature vectors, converges in the mean to $V_U$, the value function for a given control policy $U$. Dayan (1992) also shows that linear TD($\lambda$) with linearly *dependent* state representations converges, but not to $V_U$, the function that the algorithm is supposed to learn.

Despite the paucity of theoretical results, applications have shown promise. For example, Tesauro (1992) describes a system using TD($\lambda$) that learns to play championship level backgammon entirely through self-play[1]. It uses a multilayer perceptron (MLP) trained using backpropagation as a function approximator. Sofge and White (1990) describe a system that learns to improve process control with continuous state and action spaces. Neither of these applications, nor many similar applications that have been described, meet the convergence requirements of the existing theory. Yet they produce good results experimentally. We need to extend the theory of DP-based reinforcement learning to domains with continuous state and action spaces, and to algorithms that use non-linear function approximators.

Linear Quadratic Regulation (e.g., Bertsekas, 1987) is a good candidate as a first attempt in extending the theory of DP-based reinforcement learning in this manner. LQR is an important class of control problems and has a well-developed theory. LQR problems involve continuous state and action spaces, and value functions can be exactly represented by quadratic functions. The following sections review the basics of LQR theory that will be needed in this paper, describe $Q$-functions for LQR, describe the $Q$-learning algorithm used in this paper, and describe an algorithm based on $Q$-learning that is proven to converge to the optimal controller for a large class of LQR problems. We also describe a slightly different algorithm that is only *locally* convergent to the optimal $Q$-function, demonstrating one of the possible pitfalls of using a non-linear function approximator with DP-based learning.

## 2     LINEAR QUADRATIC REGULATION

Consider the deterministic, linear, time-invariant, discrete time dynamical system given by

$$
\begin{aligned}
x_{t+1} &= f(x_t, u_t) \\
&= Ax_t + Bu_t \\
u_t &= Ux_t,
\end{aligned}
$$

where $A$, $B$, and $U$ are matrices of dimensions $n \times n$, $n \times m$, and $m \times n$ respectively. $x_t$ is the state of the system at time $t$, and $u_t$ is the control input to the system at

time $t$. $U$ is a linear feedback controller. The cost at every time step is a quadratic function of the state and the control signal:

$$
\begin{aligned}
r_t &= r(x_t, u_t) \\
&= x_t' E x_t + u_t' F u_t,
\end{aligned}
$$

where $E$ and $F$ are symmetric, positive definite matrices of dimensions $n \times n$ and $m \times m$ respectively, and $x'$ denotes $x$ transpose.

The value $V_U(x_t)$ of a state $x_t$ under a given control policy $U$ is defined as the discounted sum of all costs that will be incurred by using $U$ for all times from $t$ onward, i.e., $V_U(x_t) = \sum_{i=0}^{\infty} \gamma^i r_{t+i}$, where $0 \leq \gamma \leq 1$ is the discount factor. Linear-quadratic control theory (e.g., Bertsekas, 1987) tells us that $V_U$ is a quadratic function of the states and can be expressed as $V_U(x_t) = x_t' K_U x_t$, where $K_U$ is the $n \times n$ *cost matrix* for policy $U$. The optimal control policy, $U^*$, is that policy for which the value of every state is minimized. We denote the cost matrix for the optimal policy by $K^*$.

## 3   $Q$-FUNCTIONS FOR LQR

Watkins (1989) defined the $Q$-function for a given control policy $U$ as $Q_U(x, u) = r(x, u) + \gamma V_U(f(x, u))$. This can be expressed for an LQR problem as

$$
\begin{aligned}
Q_U(x, u) &= r(x, u) + \gamma V_U(f(x, u)) \\
&= x' E x + u' F u + \gamma (Ax + Bu)' K_U (Ax + Bu) \\
&= [x, u]' \begin{bmatrix} E + \gamma A' K_U A & \gamma A' K_U B \\ \gamma B' K_U A & F + \gamma B' K_U B \end{bmatrix} [x, u],
\end{aligned} \tag{1}
$$

where $[x, u]$ is the column vector concatenation of the column vectors $x$ and $u$.

Define the parameter matrix $H_U$ as

$$
H_U = \begin{bmatrix} E + \gamma A' K_U A & \gamma A' K_U B \\ \gamma B' K_U A & F + \gamma B' K_U B \end{bmatrix} = \begin{bmatrix} H_{11} & H_{12} \\ H_{21} & H_{22} \end{bmatrix}. \tag{2}
$$

$H_U$ is a symmetric positive definite matrix of dimensions $(n + m) \times (n + m)$.

## 4   $Q$-LEARNING FOR LQR

The convergence results for $Q$-learning (Watkins & Dayan, 1992) assume a discrete time, finite-state system, and require the use of lookup-tables to represent the $Q$-function. This is not suitable for the LQR domain, where the states and actions are vectors of real numbers. Following the work of others, we will use a parameterized representation of the $Q$-function and adjust the parameters through a learning process. For example, Jordan and Jacobs (1990) and Lin (1992) use MLPs trained using backpropagation to approximate the $Q$-function. Notice that the function $Q_U$ is a quadratic function of its arguments, the state and control action, but it is a *linear* function of the quadratic combinations from the vector $[x, u]$. For example, if $x = [x_1, x_2]$, and $u = [u_1]$, then $Q_U(x, u)$ is a linear function of

the vector $[x_1^2, x_2^2, u_1^2, x_1 x_2, x_1 u_1, x_2 u_1]$. This fact allows us to use linear Recursive Least Squares (RLS) to implement $Q$-learning in the LQR domain.

There are two forms of $Q$-learning. The first is the rule Watkins described in his thesis (Watkins, 1989) . Watkins called this rule *Q-learning*, but we will refer to it as *optimizing Q-learning* because it attempts to learn the $Q$-function of the optimal policy directly. The optimizing $Q$-learning rule may be written as

$$Q_{t+1}(x_t, u_t) = Q_t(x_t, u_t) + \alpha \left[ r(x_t, u_t) + \gamma \min_a Q_t(x_{t+1}, a) - Q_t(x_t, u_t) \right], \quad (3)$$

where $Q_t$ is the $t^{\text{th}}$ approximation to $Q^{\cdot}$. The second form of $Q$-learning attempts to learn $Q_U$, the $Q$-function for some designated policy, $U$. $U$ may or may not be the policy that is actually followed during training. This *policy-based Q-learning* rule may be written as

$$Q_{t+1}(x_t, u_t) = Q_t(x_t, u_t) + \alpha \left[ r(x_t, u_t) + \gamma Q_t(x_{t+1}, U x_{t+1}) - Q_t(x_t, u_t) \right], \quad (4)$$

where $Q_t$ is the $t^{\text{th}}$ approximation to $Q_U$. Bradtke, Ydstie, and Barto (paper in preparation) show that a linear RLS implementation of the policy-based $Q$-learning rule will converge to $Q_U$ for LQR problems.

## 5   POLICY IMPROVEMENT FOR LQR

Given a policy $U_k$, how can we find an improved policy, $U_{k+1}$? Following Howard (1960) , define $U_{k+1}$ as

$$U_{k+1} x = \operatorname*{argmin}_u \left[ r(x, u) + \gamma V_{U_k}(f(x, u)) \right].$$

But equation (1) tells us that this can be rewritten as

$$U_{k+1} x = \operatorname*{argmin}_u Q_{U_k}(x, u).$$

We can find the minimizing $u$ by taking the partial derivative of $Q_{U_k}(x, u)$ with respect to $u$, setting that to zero, and solving for $u$. This yields

$$u = \underbrace{-\gamma \left( F + \gamma B' K_{U_k} B \right)^{-1} B' K_{U_k} A}_{U_{k+1}} x.$$

Using (2), $U_{k+1}$ can be written as

$$U_{k+1} = -H_{22}^{-1} H_{21}.$$

Therefore we can use the definition of the $Q$-function to compute an improved policy.

## 6   POLICY ITERATION FOR LQR

The RLS implementation of policy-based $Q$-learning (Section 4) and the policy improvement process based on $Q$-functions (Section 5) are the key elements of the policy iteration algorithm described in Figure 1. Theorem 1, proven in (Bradtke,

Ydstie, & Barto, in preparation), shows that the sequence of policies generated by this algorithm converges to the optimal policy. Standard policy iteration algorithms, such as those described by Howard (1960) for discrete time, finite state Markovian decision tasks, or by Bertsekas (1987) and Kleinman (1968) for LQR problems, require exact knowledge of the system model. *Our algorithm requires no system model. It only requires a suitably accurate estimate of $H_{U_k}$.*

**Theorem 1**: If (1) $\{A, B\}$ is controllable, (2) $U_0$ is stabilizing, and (3) the control signal, which at time step $t$ and policy iteration step $k$ is $U_k x_t$ plus some "exploration factor", is strongly persistently exciting, then there exists a number $N$ such that the sequence of policies generated by the policy iteration algorithm described in Figure 1 will converge to $U^*$ when policy updates are performed at most every $N$ time steps.

---

Initialize the $Q$-function parameters, $\hat{H}_0$.
$t = 0$, $k = 0$.
do forever {
        Initialize the Recursive Least Squares estimator.
        for i = 1 to N {
                • $u_t = U_k x_t + e_t$, where $e_t$ is the "exploration" component of the control signal.
                • Apply $u_t$ to the system, resulting in state $x_{t+1}$.
                • Define $a_{t+1} = U_k x_{t+1}$.
                • Update the $Q$-function parameters, $\hat{H}_k$ using the Recursive Least Squares implementation of the policy-based $Q$-learning rule, equation (4).
                • $t = t + 1$.
        }
        Policy improvement based on $\hat{H}_k$: $U_{k+1} = -\hat{H}_{22}^{-1}\hat{H}_{21}$
        Initialize parameters $\hat{H}_{k+1} = \hat{H}_k$.
        $k = k + 1$
}

---

Figure 1: The $Q$-function based policy iteration algorithm. It starts with the system in some initial state $x_0$ and with some stabilizing controller $U_0$. $k$ keeps track of the number of policy iteration steps. $t$ keeps track of the total number of time steps. $i$ counts the number of time steps since the last change of policy. When $i = N$, one policy improvement step is executed.

Figure 2 demonstrates the performance of the $Q$-function based policy iteration algorithm. We do not know how to characterize a persistently exciting exploratory signal for this algorithm. Experimentally, however, a random exploration signal generated from a normal distribution has worked very well, even though it does not meet condition (3) of the theorem. The system is a 20-dimensional discrete time approximation of a flexible beam supported at both ends. There is one control point. The control signal is a scalar representing acceleration to be applied at that point. $U_0$ is an arbitrarily selected stabilizing controller for the system. $x_0$ is a random

point in a neighborhood around $0 \in \mathcal{R}^{20}$. We used a normal random variable with mean 0 and variance 1 as the exploratory signal. There are 231 parameters to be estimated for this system, so we set $N = 500$, approximately twice that. Panel A of Figure 2 shows the norm of the difference between the current controller and the optimal controller. Panel B of Figure 2 shows the norm of the difference between the estimate of the $Q$-function for the current controller and the $Q$-function for the optimal controller. After only eight policy iteration steps the $Q$-function based policy iteration algorithm has converged close enough to $U^{\ast}$ and $Q^{\ast}$ that further improvements are limited by the machine precision.

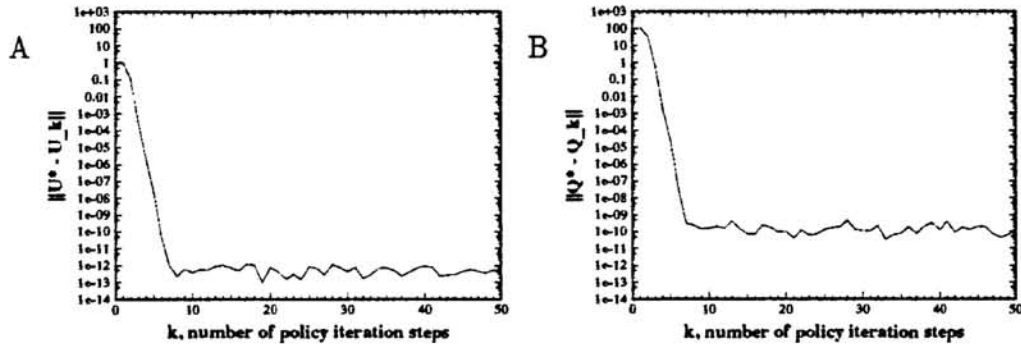

Figure 2: Performance of the $Q$-function based policy iteration algorithm on a discretized beam system.

## 7   THE OPTIMIZING $Q$-LEARNING RULE FOR LQR

Policy iteration would seem to be a slow method. It has to evaluate each policy before it can specify a new one. Why not do as Watkins' optimizing $Q$-learning rule does (equation 3), and try to learn $Q^{\ast}$ directly? Figure 3 defines this algorithm precisely. This algorithm does not update the policy actually used during training. It only updates the estimate of $Q^{\ast}$. The system is started in some initial state $x_0$ and some stabilizing controller $U_0$ is specified as the controller to be used during training.

To what will this algorithm converge, if it does converge? A fixed point of this algorithm must satisfy

$$[x, u]' \begin{bmatrix} H_{11} & H_{12} \\ H_{21} & H_{22} \end{bmatrix} [x, u] =$$

$$x'Ex + u'Eu + \gamma[Ax + Bu, a]' \begin{bmatrix} H_{11} & H_{12} \\ H_{21} & H_{22} \end{bmatrix} [Ax + Bu, a], \qquad (5)$$

where $a = -H_{22}^{-1}H_{21}(Ax+Bu)$. Equation (5) actually specifies $(n+m)(n+m+1)/2$ polynomial equations in $(n+m)(n+m+1)/2$ unknowns (remember that $H_U$ is symmetric). We know that there is at least one solution, that corresponding to the optimal policy, but there may be other solutions as well.

As an example of the possibility of multiple solutions, consider the 1-dimensional system with $A = B = E = F = [1]$ and $\gamma = 0.9$. Substituting these values into

Initialize the $Q$-function parameters, $\hat{H}_0$.
Initialize Recursive Least Squares estimator.
$t = 0$.
do forever {

- $u_t = U_0 x_t + e_t$, where $e_t$ is the "exploration" component of the control signal.
- Apply $u_t$ to the system, resulting in state $x_{t+1}$.
- Define $a_{t+1} = -\hat{H}_{22}^{-1} \hat{H}_{21} x_{t+1}$.
- Update the $Q$-function parameters, $\hat{H}_t$, using the Recursive Least Squares implementation of the optimizing $Q$-learning rule, equation (3).
- $t = t + 1$.

}

Figure 3: The optimizing $Q$-learning rule in the LQR domain. $U_0$ is the policy followed during training. $t$ keeps track of the total number of time steps.

equation (5) and solving for the unknown parameters yields two solutions. They are

$$\begin{bmatrix} 2.4296 & 1.4296 \\ 1.4296 & 2.4296 \end{bmatrix} \text{ and } \begin{bmatrix} 0.3704 & -0.6296 \\ -0.6296 & 0.3704 \end{bmatrix}.$$

The first solution is $Q^*$. The second solution, if used to define an "improved" policy as describe in Section 5, results in a destablizing controller. This is certainly not a desirable result. Experiments show that the algorithm in Figure 3 will converge to either of these solutions if the initial parameter estimates are close enough to that solution. Therefore, this method of using Watkins' $Q$-learning rule directly on an LQR problem will not necessarily converge to the optimal $Q$-function.

## 8   CONCLUSIONS

In this paper we take a first step toward extending the theory of DP-based reinforcement learning to domains with continuous state and action spaces, and to algorithms that use non-linear function approximators. We concentrate on the problem of Linear Quadratic Regulation. We describe a policy iteration algorithm for LQR problems that is proven to converge to the optimal policy. In contrast to standard methods of policy iteration, it does not require a system model. It only requires a suitably accurate estimate of $H_{U_k}$. This is the first result of which we are aware showing convergence of a DP-based reinforcement learning algorithm in a domain with continuous states and actions. We also describe a straightforward implementation of the optimizing $Q$-learning rule in the LQR domain. This algorithm is only locally convergent to $Q^*$. This result demonstrates that we cannot expect the theory developed for finite-state systems using lookup-tables to extend to continuous state systems using parameterized function representations.

The convergence proof for the policy iteration algorithm described in this paper requires exact matching between the form of the $Q$-function for LQR problems and the form of the function approximator used to learn that function. Future work will explore convergence of DP-based reinforcement learning algorithms when applied to non-linear systems for which the form of the $Q$-functions is unknown.

## Acknowledgements

The author thanks Andrew Barto, B. Erik Ydstie, and the ANW group for their contributions to these ideas. This work was supported by the Air Force Office of Scientific Research, Bolling AFB, under Grant AFOSR-89-0526 and by the National Science Foundation under Grant ECS-8912623.

## Footnotes

[1]Backgammon can be viewed as a Markovian decision task.

## References

[1] D. P. Bertsekas. *Dynamic Programming: Deterministic and Stochastic Models.* Prentice Hall, Englewood Cliffs, NJ, 1987.

[2] S. J. Bradtke, B. E. Ydstie, and A. G. Barto. Convergence to optimal cost of adaptive policy iteration. In preparation.

[3] P. Dayan. The convergence of TD($\lambda$) for general $\lambda$. *Machine Learning*, 1992.

[4] R. A. Howard. *Dynamic Programming and Markov Processes.* John Wiley & Sons, Inc., New York, 1960.

[5] M. I. Jordan and R. A. Jacobs. Learning to control an unstable system with forward modeling. In *Advances in Neural Information Processing Systems 2.* Morgan Kaufmann Publishers, San Mateo, CA, 1990.

[6] D. L. Kleinman. On an iterative technique for Riccati equation computations. *IEEE Transactions on Automatic Control*, pages 114–115, February 1968.

[7] L.-J. Lin. Self-improving reactive agents based on reinforcement learning, planning and teaching. *Machine Learning*, 1992.

[8] D. A. Sofge and D. A. White. Neural network based process optimization and control. In *Proceedings of the $29^{th}$ IEEE Conference on Decision and Control*, Honolulu, Hawaii, December 1990.

[9] R. S. Sutton. Learning to predict by the method of temporal differences. *Machine Learning*, 3:9–44, 1988.

[10] G. J. Tesauro. Practical issues in temporal difference learning. *Machine Learning*, 8(3/4):257–277, May 1992.

[11] C. J. C. H. Watkins. *Learning from Delayed Rewards.* PhD thesis, Cambridge University, Cambridge, England, 1989.

[12] C. J. C. H. Watkins and P. Dayan. Q-learning. *Machine Learning*, 1992.

[13] P. J. Werbos. Building and understanding adaptive systems: A statistical/numerical approach to factory automation and brain research. *IEEE Transactions on Systems, Man, and Cybernetics*, 17(1):7–20, 1987.